# Interpretation of Artificial Neural Networks: Mapping Knowledge-Based Neural Networks into Rules

Geoffrey Towell         Jude W. Shavlik
Computer Sciences Department
University of Wisconsin
Madison, WI 53706

## Abstract

We propose and empirically evaluate a method for the extraction of expert-comprehensible rules from trained neural networks. Our method operates in the context of a three-step process for learning that uses rule-based domain knowledge in combination with neural networks. Empirical tests using real-worlds problems from molecular biology show that the rules our method extracts from trained neural networks: closely reproduce the accuracy of the network from which they came, are superior to the rules derived by a learning system that directly refines symbolic rules, and are expert-comprehensible.

## 1 Introduction

Artificial neural networks (ANNs) have proven to be a powerful and general technique for machine learning [1, 11]. However, ANNs have several well-known shortcomings. Perhaps the most significant of these shortcomings is that determining why a trained ANN makes a particular decision is all but impossible. Without the ability to explain their decisions, it is hard to be confident in the reliability of a network that addresses a real-world problem. Moreover, this shortcoming makes it difficult to transfer the information learned by a network to the solution of related problems. Therefore, methods for the extraction of comprehensible, symbolic rules from trained networks are desirable.

Our approach to understanding trained networks uses the three-link chain illustrated by Figure 1. The first link inserts domain knowledge, which need be neither complete nor correct, into a neural network using KBANN [13] — see Section 2. (Networks created using KBANN are called KNNs.) The second link trains the KNN using a set of classified

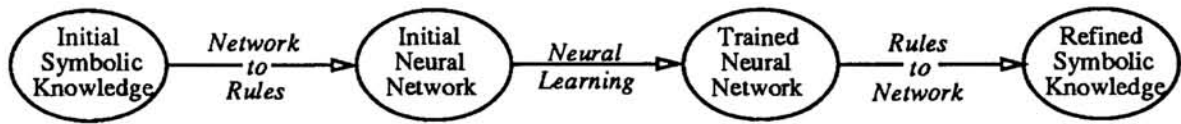

Figure 1: Rule refinement using neural networks.

training examples and standard neural learning methods [9]. The final link extracts rules from trained KNNs. Rule extraction is an extremely difficult task for arbitrarily-configured networks, but is somewhat less daunting for KNNs due to their initial comprehensibility. Our method (described in Section 3) takes advantage of this property to efficiently extract rules from trained KNNs.

Significantly, when evaluated in terms of the ability to correctly classify examples not seen during training, our method produces rules that are equal or superior to the networks from which they came (see Section 4). Moreover, the extracted rules are superior to the rules resulting from methods that act *directly* on the rules (rather than their re-representation as a neural network). Also, our method is superior to the most widely-published algorithm for the extraction of rules from general neural networks.

## 2   The KBANN Algorithm

The KBANN algorithm translates symbolic domain knowledge into neural networks; defining the topology and connection weights of the networks it creates. It uses a knowledge base of domain-specific inference rules to define what is initially known about a topic. A detailed explanation of this rule-translation appears in [13].

As an example of the KBANN method, consider the sample domain knowledge in Figure 2a that defines membership in category A. Figure 2b represents the hierarchical structure of these rules: solid and dotted lines represent necessary and prohibitory dependencies, respectively. Figure 2c represents the KNN that results from the translation into a neural network of this domain knowledge. Units X and Y in Figure 2c are introduced into the KNN to handle the disjunction in the rule set. Otherwise, each unit in the KNN corresponds to a consequent or an antecedent in the domain knowledge. The thick lines in Figure 2c represent heavily-weighted links in the KNN that correspond to dependencies in the domain knowledge. The thin lines represent the links added to the network to allow refinement of the domain knowledge. Weights and biases in the network are set so that, prior to learning, the network's response to inputs is exactly the same as the domain knowledge.

This example illustrates the two principal benefits of using KBANN to initialize KNNs. First, the algorithm indicates the features that are believed to be important to an example's classification. Second, it specifies important derived features, thereby guiding the choice of the number and connectivity of *hidden units*.

## 3   Rule Extraction

Almost every method of rule extraction makes two assumptions about networks. First, that training does not significantly shift the meaning of units. By making this assumption, the methods are able to attach labels to rules that correspond to terms in the domain knowledge

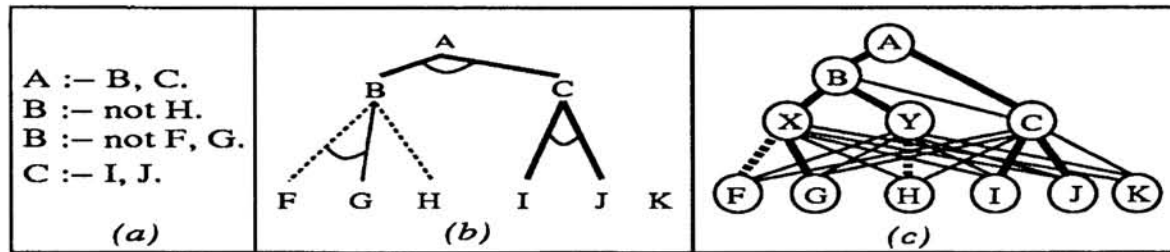

Figure 2: Translation of domain knowledge into a KNN.

upon which the network is based. These labels enhance the comprehensibility of the rules. The second assumption is that the units in a trained KNN are always either active ($\approx$ 1) or inactive ($\approx$ 0). Under this assumption each non-input unit in a trained KNN can be treated as a Boolean rule. Therefore, the problem for rule extraction is to determine the situations in which the "rule" is true. Examination of trained KNNs validates both of these assumptions.

Given these assumptions, the simplest method for extracting rules we call the SUBSET method. This method operates by exhaustively searching for subsets of the links into a unit such that the sum of the weights of the links in the subset guarantees that the total input to the unit exceeds its bias. In the limit, SUBSET extracts a set of rules that reproduces the behavior of the network. However, the combinatorics of this method render it impossible to implement. Heuristics can be added to reduce the complexity of the search at some cost in the accuracy of the resulting rules. Using heuristic search, SUBSET tends to produce repetitive rules whose preconditions are difficult to interpret. (See [10] or [2] for more detailed explanations of SUBSET.)

Our algorithm, called NOFM, addresses both the combinatorial and presentation problems inherent to the SUBSET algorithm. It differs from SUBSET in that it explicitly searches for rules of the form: ``If ($N$ of these $M$ antecedents are true) ...'' This method arose because we noticed that rule sets discovered by the SUBSET method often contain N-of-M style concepts. Further support for this method comes from experiments that indicate neural networks are good at learning N-of-M concepts [1] as well as experiments that show a bias towards N-of-M style concepts is useful [5]. Finally, note that purely conjunctive rules result if $N = M$, while a set of disjunctive rules results when $N = 1$; hence, using N-of-M rules does not restrict generality.

The idea underlying NOFM (summarized in Table 1) is that individual antecedents (links) do not have unique importance. Rather, groups of antecedents form equivalence classes in which each antecedent has the same importance as, and is interchangeable with, other members of the class. This equivalence-class idea allows NOFM to consider groups of links without worrying about particular links within the group. Unfortunately, training using backpropagation does not naturally bunch links into equivalence classes. Hence, the first step of NOFM groups links into equivalence classes.

This grouping can be done using standard clustering methods [3] in which clustering is stopped when no clusters are closer than a user-set distance (we use 0.25). After clustering, the links to the unit in the upper-right corner of Figure 3 form two groups, one of four links with weight near one and one of three links with weight near six. (The effect of this grouping is very similar to the training method suggested by Nowlan and Hinton [7].)

Table 1: The NoFM algorithm for rule extraction.

(1)    With each hidden and output unit, form groups of similarly-weighted links.

(2)    Set link weights of all group members to the average of the group.

(3)    Eliminate any groups that do not affect whether the unit will be active or inactive.

(4)    Holding all links weights constant, optimize biases of hidden and output units.

(5)    Form a single rule for each hidden and output unit. The rule consists of a threshold given by the bias and weighted antecedents specified by remaining links.

(6)    Where possible, simplify rules to eliminate spperfluous weights and thresholds.

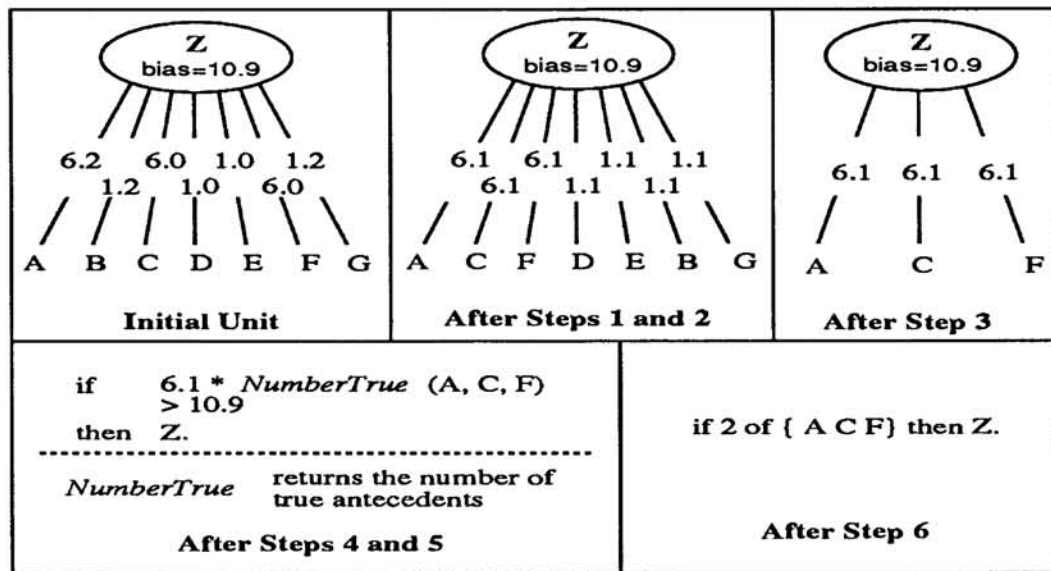

Figure 3: Rule extraction using NoFM.

Once the groups are formed, the procedure next attempts to identify and eliminate groups that do not contribute to the calculation of the consequent. In the extreme case, this analysis is trivial; clusters can be eliminated solely on the basis of their weight. In Figure 3 no combination of the cluster of links with weight 1.1 can cause the summed weights to exceed the bias on unit Z. Hence, links with weight 1.1 are eliminated from Figure 3 after step 3.

More often, the assessment of a cluster's utility uses heuristics. The heuristic we use is to scan each training example and determine which groups can be eliminated while leaving the example correctly categorized. Groups not required by any example are eliminated.

With unimportant groups eliminated, the next step of the procedure is to optimize the bias on each unit. Optimization is required to adjust the network so that it accurately reflects the assumption that units are boolean. This can be done by freezing link weights (so that the groups stay intact) and retraining the bias terms in the network.

After optimization, rules are formed that simply re-express the network. Note that these rules are considerable simpler than the trained network; they have fewer antecedents and those antecedents tend to be in a few weight classes.

Finally, rules are simplified whenever possible to eliminate the weights and thresholds. Simplification is accomplished by a scan of each restated rule to determine combinations of

clusters that exceed the threshold. In Figure 3 the result of this scan is a single N-of-M style rule. When a rule has more than one cluster, this scan may return multiple combinations each of which has several N-of-M predicates. In such cases, rules are left in their original form of weights and a threshold.

# 4    Experiments in Rule Extraction

This section presents a set of experiments designed to determine the relative strengths and weaknesses of the two rule-extraction methods described above. Rule-extraction techniques are compared using two measures: *quality*, which is measured both by the accuracy of the rules; and *comprehensibility* which is approximated by analysis of extracted rule sets.

## 4.1    Testing Methodology

Following Weiss and Kulikowski [14], we use repeated 10-fold cross-validation[1] for testing learning on two tasks from molecular biology: promoter recognition [13] and splice-junction determination [6]. Networks are trained using the *cross-entropy*. Following Hinton's [4] suggestion for improved network interpretability, all weights "decay" gently during training.

## 4.2    Accuracy of Extracted Rules

Figure 4 addresses the issue of the accuracy of extracted rules. It plots percentage of errors on the testing and training sets, averaged over eleven repetitions of 10-fold cross-validation, for both the promoter and splice-junction tasks. For comparison, Figure 4 includes the accuracy of the trained KNNs prior to rule extraction (the bars labeled "Network"). Also included in Figure 4 is the accuracy of the EITHER system, an "all symbolic" method for the empirical adaptation of rules [8]. (EITHER has not been applied to the splice-junction problem.)

The initial rule sets for promoter recognition and splice-junction determination correctly categorized 50% and 61%, respectively, of the examples. Hence, each of the systems plotted in Figure 4 improved upon the initial rules. Comparing only the systems that result in refined rules, the NOFM method is the clear winner. On training examples, the error rate for rules extracted by NOFM is slightly worse than EITHER but superior to the rules extracted using SUBSET. On the testing examples the NOFM rules are more accurate than both EITHER and SUBSET. (One-tailed, paired-sample *t*-tests indicate that for both domains the NOFM rules are superior to the SUBSET rules with 99.5% confidence.)

Perhaps the most significant result in this paper is that, on the testing set, the error rate of the NOFM rules is equal or superior to that of the networks from which the rules were extracted. Conversely, the error rate of the SUBSET rules on testing examples is statistically worse than the networks in both problem domains. The discussion at the end of this paper

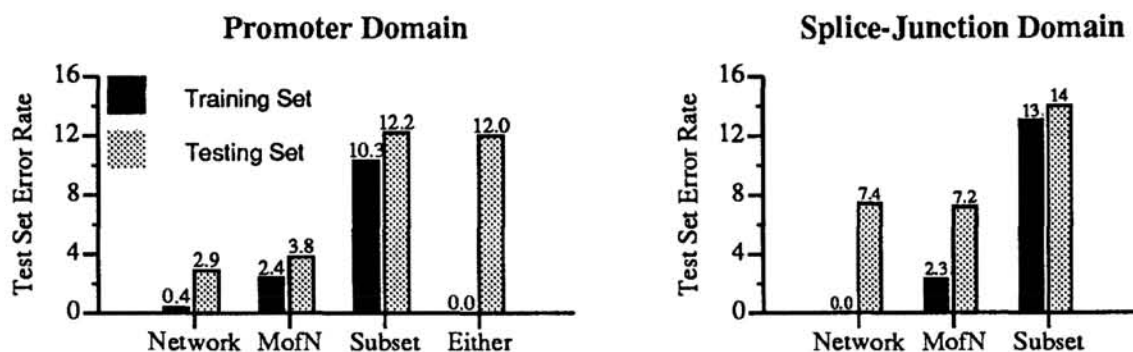

Figure 4: Error rates of extracted rules.

analyses the reasons why NoFM's rules can be superior to the networks from which they came.

## 4.3  Comprehensibility

To be useful, the extracted rules must not only be accurate, they also must be understandable. To assess rule comprehensibility, we looked at rule sets extracted by the NoFM method. Table 3 presents the rules extracted by NoFM for promoter recognition. The rules extracted by NoFM for splice-junction determination are not shown because they have much the same character as those of the promoter domain.

While Table 3 is somewhat murky, it is vastly more comprehensible than the network of 3000 links from which it was extracted. Moreover, the rules in this table can be rewritten in a form very similar to one used in the biological community [12], namely weight matrices.

One major pattern in the extracted rules is that the network learns to disregard a major portion of the initial rules. These same rules are dropped by other rule-refinement systems (e.g., EITHER). This suggests that the deletion of these rules is not merely an artifact of NoFM, but instead reflects an underlying property of the data. Hence, we demonstrate that machine learning methods can provide valuable evidence about biological theories.

Looking beyond the dropped rules, the rules NoFM extracts confirm the importance of the bases identified in the initial rules (Table 2). However, whereas the initial rules required matching every base, the extracted rules allow a less than perfect match. In addition, the extracted rules point to places in which changes to the sequence are important. For instance, in the first *minus10* rule, a 'T' in position 11 is a strong indicator that the rule is true. However, replacing the 'T' with either a 'G' or an 'A' prevents the rule from being satisfied.

## 5  Discussion and Conclusions

Our results indicate that the NoFM method not only can extract meaningful, symbolic rules from trained KNNs, the extracted rules can be superior at classifying examples not seen during training to the networks from which they came. Additionally, the NoFM method produces rules whose accuracy is substantially better than EITHER, an approach that directly modifies the initial set of rules [8]. While the rule set produced by the NoFM algorithm is

Table 2: Partial set of original rules for promoter-recognition.

```
promoter        :- contact, conformation.
contact         :- minus-35, minus-10.
minus-35        :- @-37 'CTTGAC'.     --- three additional rules
minus-10        :- @-14 'TATAAT'.     --- three additional rules
conformation    :-  @-45 'AA--A'.     --- three additional rules
```

Examples are 57 base-pair long strands of DNA. Rules refer to bases by stating a sequence location followed by a subsequnce. So, @-37 'CT' indicates a 'C' in position -37 and a 'T' in position -36.

Table 3: Promoter rules NoFM extracts.

```
Promoter :- Minus35, Minus10.

Minus-35                                    Minus-10 :- 2 of @-14 '---CA---T' and
  :-10 < 4.0 * nt(@-37 '--TTGAT-') +                not 1 of @-14 '---RB---S'.
         1.5 * nt(@-37 '----TCC-') +        Minus-10
         0.5 * nt(@-37 '---MC---') -          :-10 < 3.0 * nt(@-14 '--TAT--T-') +
         1.5 * nt(@-37 '--GGAGG-').                 1.8 * nt(@-14 '-----GA--') +
Minus-35                                            0.7 * nt(@-14 '----GAT--') -
  :-10 < 5.0 * nt(@-37 '--T-G--A') +                0.7 * nt(@-14 '--GKCCCS-').
         3.1 * nt(@-37 '---GT---') +         Minus-10
         1.9 * nt(@-37 '-----C-CT') +          :-10 < 3.8 * nt(@-14 '--TA-A-T-') +
         1.5 * nt(@-37 '---C--A-') -                 3.0 * nt(@-14 '--G--C---') +
         1.5 * nt(@-37 '-------GC') -                1.0 * nt(@-14 '---T---A-') -
         1.9 * nt(@-37 '--CAW---') -                 1.0 * nt(@-14 '--CS-G-S-') -
         3.1 * nt(@-37 '--A----C').                  3.0 * nt(@-14 '--A--T---').
Minus-35 :- @-37 '-C-TGAC-'.                 Minus-10 :-  @-14 '-TAWA-T--'.
Minus-35 :- @-37 '--TTD-CA'.
```

"nt()" returns the number of enclosed in the parentheses antecedents that match the given sequence. So, nt(@-14 '- - - C - - G - -') would return 1 when matched against the sequence @-14 'AAACAAAAA'.

Table 4: Standard nucleotide ambiguity codes.

| Code | Meaning | Code | Meaning | Code | Meaning | Code | Meaning |
|------|---------|------|---------|------|---------|------|---------|
| M | A or C | R | A or G | W | A or T | S | C or G |
| K | G or T | D | A or G or T | B | C or G or T | | |

slightly larger than that produced by EITHER, the sets of rules produced by both of these algorithms is small enough to be easily understood. Hence, although weighing the tradeoff between accuracy and understandability is problem and user-specific, the NOFM approach combined with KBANN offers an appealing mixture.

The superiority of the NOFM rules over the networks from which they are extracted may occur because the rule-extraction process reduces overfitting of the training examples. The principle evidence in support of this hypothesis is that the difference in ability to correctly categorize testing and training examples is smaller for NOFM rules than for trained KNNs. Thus, the rules extracted by NOFM sacrifice some training set accuracy to achieve higher testing set accuracy.

Additionally, in earlier tests this effect was more pronounced; the NOFM rules were superior to the networks from which they came on both datasets (with 99according to a one-tailed $t$-test). Modifications to training to reduce overfitting improved generalization by networks without significantly affecting NOFM's rules. The result of the change in training method is that the differences between the network and NOFM are not statistically significant in either dataset. However, the result is significant in that it supports the overfitting hypothesis.

In summary, the NoFM method extracts accurate, comprehensible rules from trained KNNs. The method is currently limited to KNNs; randomly-configured networks violate its assumptions. New training methods [7] may broaden the applicability of the method. Even without different methods for training, our results show that NoFM provides a mechanism through which networks can make expert comprehensible explanations of their behavior. In addition, the extracted rules allow for the transfer of learning to the solution of related problems.

## Acknowledgments

This work is partially supported by Office of Naval Research Grant N00014-90-J-1941, National Science Foundation Grant IRI-9002413, and Department of Energy Grant DE-FG02-91ER61129.

## Footnotes

[1] In $N$-fold cross-validation, the set of examples is partitioned into $N$ sets of equal size. Networks are trained using $N - 1$ of the sets and tested using the remaining set. This procedure is repeated $N$ times so that each set is used as the testing set once. We actually used only $N - 2$ of the sets for training. One set was used for testing and the other to stop training to prevent overfitting of the training set.

## References

[1] D. H. Fisher and K. B. McKusick. An empirical comparison of ID3 and back-propagation. In *Proceedings of the Eleventh International Joint Conference on Artificial Intelligence*, pages 788–793, Detroit, MI, August 1989.

[2] L. M. Fu. Rule learning by searching on adapted nets. In *Proceedings of the Ninth National Conference on Artificial Intelligence*, pages 590–595, Anaheim, CA, 1991.

[3] J. A. Hartigan. *Clustering Algorithms*. Wiley, New York, 1975.

[4] G. E. Hinton. Connectionist learning procedures. *Artificial Intelligence*, 40:185–234, 1989.

[5] P. M. Murphy and M. J. Pazzani. ID2-of-3: Constructive induction of N-of-M concepts for discriminators in decision trees. In *Proceedings of the Eighth International Machine Learning Workshop*, pages 183–187, Evanston, IL, 1991.

[6] M. O. Noordewier, G. G. Towell, and J. W. Shavlik. Training knowledge-based neural networks to recognize genes in DNA sequences. In *Advances in Neural Information Processing Systems*, 3, Denver, CO, 1991. Morgan Kaufmann.

[7] S. J. Nowlan and G. E. Hinton. Simplifying neural networks by soft weight-sharing. In *Advances in Neural Information Processing Systems*, 4, Denver, CO, 1991. Morgan Kaufmann.

[8] D. Ourston and R. J. Mooney. Changing the rules: A comprehensive approach to theory refinement. In *Proceedings of the Eighth National Conference on Artificial Intelligence*, pages 815–820, Boston, MA, Aug 1990.

[9] D. E. Rumelhart, G. E. Hinton, and R. J. Williams. Learning internal representations by error propagation. In D. E. Rumelhart and J. L. McClelland, editors, *Parallel Distributed Processing: Explorations in the microstructure of cognition. Volume 1: Foundations*, pages 318–363. MIT Press, Cambridge, MA, 1986.

[10] K. Saito and R. Nakano. Medical diagnostic expert system based on PDP model. In *Proceedings of IEEE International Conference on Neural Networks*, volume 1, pages 255–262, 1988.

[11] J. W. Shavlik, R. J. Mooney, and G. G. Towell. Symbolic and neural net learning algorithms: An empirical comparison. *Machine Learning*, 6:111–143, 1991.

[12] G. D. Stormo. Consensus patterns in DNA. In *Methods in Enzymology*, volume 183, pages 211–221. Academic Press, Orlando, FL, 1990.

[13] G. G. Towell, J. W. Shavlik, and M. O. Noordewier. Refinement of approximately correct domain theories by knowledge-based neural networks. In *Proceedings of the Eighth National Conference on Artificial Intelligence*, pages 861–866, Boston, MA, 1990.

[14] S. M. Weiss and C. A. Kulikowski. *Computer Systems that Learn*. Morgan Kaufmann, San Mateo, CA, 1990.
